# A Polynomial-time Form of Robust Regression

**Yaoliang Yu, Özlem Aslan and Dale Schuurmans**
Department of Computing Science, University of Alberta, Edmonton AB T6G 2E8, Canada
{yaoliang,ozlem,dale}@cs.ualberta.ca

## Abstract

Despite the variety of robust regression methods that have been developed, current regression formulations are either NP-hard, or allow unbounded response to even a single leverage point. We present a general formulation for robust regression—Variational M-estimation—that unifies a number of robust regression methods while allowing a tractable approximation strategy. We develop an estimator that requires only polynomial-time, while achieving certain robustness and consistency guarantees. An experimental evaluation demonstrates the effectiveness of the new estimation approach compared to standard methods.

## 1 Introduction

It is well known that outliers have a detrimental effect on standard regression estimators. Even a single erroneous observation can arbitrarily affect the estimates produced by methods such as least squares. Unfortunately, outliers are prevalent in modern data analysis, as large data sets are automatically gathered without the benefit of manual oversight. Thus the need for regression estimators that are both scalable and robust is increasing.

Although the field of robust regression is well established, it has not considered computational complexity analysis to be one of its central concerns. Consequently, none of the standard regression estimators in the literature are both robust and tractable, even in a weak sense: it has been shown that standard robust regression formulations with non-zero breakdown are NP-hard [1, 2], while any estimator based on minimizing a convex loss cannot guarantee bounded response to even a single leverage point [3] (definitions given below). Surprisingly, there remain no standard regression formulations that guarantee both polynomial run-time with bounded response to even single outliers.

It is important to note that robustness and tractability can be achieved under restricted conditions. For example, if the domain is *bounded*, then any estimator based on minimizing a convex and Lipschitz-continuous loss achieves high breakdown [4]. Such results have been extended to kernel-based regression under the analogous assumption of a bounded kernel [5, 6]. Unfortunately, these results can no longer hold when the domain or kernel is *unbounded*: in such a case arbitrary leverage can occur [4, 7] and no (non-constant) convex loss, even Lipschitz-continuous, can ensure robustness against even a single outlier [3]. Our main motivation therefore is to extend these existing results to the case of an unbounded domain. Unfortunately, the inapplicability of convex losses in this situation means that computational tractability becomes a major challenge, and new computational strategies are required to achieve tractable robust estimators.

The main contribution of this paper is to develop a new robust regression strategy that can guarantee both polynomial run-time and bounded response to individual outliers, including leverage points. Although such an achievement is modest, it is based on two developments of interest. The first is a general formulation of adaptive M-estimation, *Variational M-estimation*, that unifies a number of robust regression formulations, including convex and bounded M-estimators with certain subset-selection estimators such as Least Trimmed Loss [7]. By incorporating Tikhonov regularization, these estimators can be extended to reproducing kernel Hilbert spaces (RKHSs). The second development is a convex relaxation scheme that ensures bounded outlier influence on the final estimator.

The overall estimation procedure is guaranteed to be tractable, robust to single outliers with unbounded leverage, and consistent under non-trivial conditions. An experimental evaluation of the proposed estimator demonstrates effective performance compared to standard robust estimators.

The closest previous works are [8], which formulated variational representations of certain robust losses, and [9], which formulated a convex relaxation of bounded loss minimization. Unfortunately, [8] did not offer a general characterization, while [9] did not prove their final estimator was robust, nor was any form of consistency established. The formulation we present in this paper generalizes [8] while the convex relaxation scheme we propose is simpler and tighter than [9]; we are thus able to establish non-trivial forms of both robustness and consistency while maintaining tractability.

There are many other notions of "robust" estimation in the machine learning literature that do not correspond to the specific notion being addressed in this paper. Work on "robust optimization" [10–12], for example, considers minimizing the worst case loss achieved given bounds on the maximum data deviation that will be considered. Such results are not relevant to the present investigation because we explicitly do not bound the magnitude of the outliers. Another notion of robustness is algorithmic stability under leave-one-out perturbation [13], which analyzes specific learning procedures rather than describing how a stable algorithm might be generally achieved.

## 2   Preliminaries

We start by considering the standard linear regression model

$$y \;=\; \mathbf{x}^T \boldsymbol{\theta}^* + u \tag{1}$$

where $\mathbf{x}$ is an $\mathbb{R}^p$-valued random variable, $u$ is a real-valued random noise term, and $\boldsymbol{\theta}^* \in \Theta \subseteq \mathbb{R}^p$ is an unknown deterministic parameter vector. Assume we are given a sample of $n$ independent identically distributed (i.i.d.) observations represented by an $n \times p$ matrix $X$ and an $n \times 1$ vector $\mathbf{y}$, where each row $X_{i:}$ is drawn from some unknown marginal probability measure $P_{\mathbf{x}}$, and $y_i$ are generated according to (1). Our task is to estimate the unknown deterministic parameter $\boldsymbol{\theta}^* \in \Theta$. Clearly, this is a well-studied problem in statistics and machine learning. If the noise distribution has a known density $p(\cdot)$, then a standard estimator is given by maximum likelihood

$$\hat{\boldsymbol{\theta}}_{ML} \;\in\; \arg\min_{\boldsymbol{\theta}\in\Theta} \tfrac{1}{n}\sum_{i=1}^{n} -\log p(y_i - X_{i:}\boldsymbol{\theta}) \;=\; \arg\min_{\boldsymbol{\theta}\in\Theta} \tfrac{1}{n}\sum_{i=1}^{n} -\log p(r_i), \tag{2}$$

where $r_i = y_i - X_{i:}\boldsymbol{\theta}$ is the $i$th residual. When the noise distribution is unknown, one can replace the negative log-likelihood with a *loss function* $\rho(\cdot)$ and use the estimator

$$\hat{\boldsymbol{\theta}}_M \;\in\; \arg\min_{\boldsymbol{\theta}\in\Theta} \tfrac{1}{n}\mathbf{1}^T \boldsymbol{\rho}(\mathbf{y} - X\boldsymbol{\theta}), \tag{3}$$

where $\boldsymbol{\rho}(\mathbf{r})$ denotes the vector of losses obtained by applying the loss componentwise to each residual, hence $\mathbf{1}^T\boldsymbol{\rho}(\mathbf{r}) = \sum_{i=1}^{n}\rho(r_i)$. Such a procedure is known as $M$-estimation in the robust statistics literature, and empirical risk minimization in the machine learning literature.[1]

Although uncommon in robust regression, it is conventional in machine learning to include a regularizer. In particular we will use Tikhonov ("ridge") regularization by adding a squared penalty

$$\hat{\boldsymbol{\theta}}_{MR} \;\in\; \arg\min_{\boldsymbol{\theta}\in\Theta} \tfrac{1}{n}\mathbf{1}^T \boldsymbol{\rho}(\mathbf{y} - X\boldsymbol{\theta}) + \tfrac{\lambda}{2}\|\boldsymbol{\theta}\|_2^2 \qquad \text{for } \lambda \geq 0, \tag{4}$$

The significance of Tikhonov regularization is that it ensures $\hat{\boldsymbol{\theta}}_{MR} = X^T\boldsymbol{\alpha}$ for some $\boldsymbol{\alpha} \in \mathbb{R}^n$ [14]. More generally, under Tikhonov regularization, the regression problem can be conveniently expressed in a reproducing kernel Hilbert space (RKHS). If we let $\mathcal{H}$ denote the RKHS corresponding to positive semidefinite kernel $\kappa : \mathcal{X} \times \mathcal{X} \to \mathbb{R}$, then $f(x) = \langle \kappa(x,\cdot), f \rangle_{\mathcal{H}}$ for any $f \in \mathcal{H}$ by the reproducing property [14, 15]. We consider the generalized regression model

$$y \;=\; f^*(x) + u \tag{5}$$

where $x$ is an $\mathcal{X}$-valued random variable, $u$ is a real-valued random noise term as above, and $f^* \in \mathcal{H}$ is an unknown deterministic function. Given a sample of $n$ i.i.d. observations $(x_1, y_1), ..., (x_n, y_n)$,

where each $x_i$ is drawn from some unknown marginal probability measure $P_x$, and $y_i$ are generated according to (5),[2] the task is then to estimate the unknown deterministic function $f^* \in \mathcal{H}$. To do so we can express the estimator (4) more generally as

$$\hat{f}_{MR} \quad \in \quad \arg\min_{f \in \mathcal{H}} \frac{1}{n} \sum_{i=1}^{n} \rho(y_i - f(x_i)) + \frac{\lambda}{2} \|f\|_{\mathcal{H}}^2. \tag{6}$$

By the representer theorem [14], the solution to (6) can be expressed by $\hat{f}_{MR}(x) = \sum_{i=1}^{n} \alpha_i \kappa(x_i, x)$ for some $\boldsymbol{\alpha} \in \mathbb{R}^n$, and therefore (6) can be recovered by solving the finite dimensional problem

$$\hat{\boldsymbol{\alpha}}_{MR} \quad \in \quad \arg\min_{\boldsymbol{\alpha}} \frac{1}{n} \mathbf{1}^T \boldsymbol{\rho}(\mathbf{y} - K\boldsymbol{\alpha}) + \frac{\lambda}{2} \boldsymbol{\alpha}^T K \boldsymbol{\alpha} \quad \text{such that } K_{ij} = \kappa(x_i, x_j). \tag{7}$$

Our interest is understanding the tractability, robustness and consistency aspects of such estimators.

***Consistency:*** Much is known about the consistency properties of estimators expressed as regularized empirical risk minimizers. For example, the ML-estimator (2) and the $M$-estimator (3) are both known to be parameter consistent under general conditions [16].[3] The regularized $M$-estimator in RKHSs (6), is loss consistent under some general assumptions on the kernel, loss and training distribution.[4] Furthermore, a weak form of $f$-consistency has also established in [6]. For bounded kernel and bounded Lipschitz losses, one can similarly prove the loss consistency of the regularized $M$-estimator (6) (in RKHS). See Appendix C.1 of the supplement for more discussion.

Generally speaking, any estimator that can be expressed as a regularized empirical loss minimization is consistent under "reasonable" conditions. That is, one can consider regularized loss minimization to be a (generally) sound principle for formulating regression estimators, at least from the perspective of consistency. However, this is no longer the case when we consider robustness and tractability; here sharp distinctions begin to arise within this class of estimators.

***Robustness:*** Although robustness is an intuitive notion, it has not been given a unique technical definition in the literature. Several definitions have been proposed, with distinct advantages and disadvantages [4]. Some standard definitions consider the asymptotic invariance of estimators to an infinitesimal but arbitrary perturbation of the underlying distribution, e.g. the influence function [4, 17]. Although these analyses can be useful, we will focus on finite sample notions of robustness since these are most related to concerns of computational tractability. In particular, we focus on the following definition related to the *finite sample breakdown point* [18, 19].

**Definition 1** (Bounded Response). *Assuming the parameter set $\Theta$ is metrizable, an estimator has bounded response if for any finite data sample its output remains in a bounded interior subset of the closed parameter set $\Theta$ (or respectively $\mathcal{H}$), no matter how a* single *observation pair is perturbed.*

This is a much weaker definition than having a non-zero breakdown point: a breakdown of $\epsilon$ requires that bounded response be guaranteed when any $\epsilon$ fraction of the data is perturbed arbitrarily. Bounded response is obviously a far more modest requirement. However, importantly, the definition of bounded response allows the possibility of arbitrary *leverage*; that is, no bound is imposed on the magnitude of a perturbed input (i.e. $\|\mathbf{x}_1\| \to \infty$ or $\kappa(x_1, x_1) \to \infty$). Surprisingly, we find that even such a weak robustness property is difficult to achieve while retaining computational tractability.

***Computational Dilemma:*** The goals of robustness and computational tractability raise a dilemma: it is easy to achieve robustness (i.e. bounded response) or tractability (i.e. polynomial run-time) in a consistent estimator, but apparently not both.

Consider, for example, using a *convex* loss function. These are the best known class of functions that admit computationally efficient polynomial-time minimization [20] (see also [21] ). It is sufficient that the objective be polynomial-time evaluable, along with its first and second derivatives,

and that the objective be *self-concordant* [20].[5] Since a Tikhonov regularizer is automatically self-concordant, the minimization problems outlined above can all be solved in polynomial time with Newton-type algorithms, provided $\rho(r)$, $\rho'(r)$, and $\rho''(r)$ can all be evaluated in polynomial time for a self-concordant $\rho$ [22, Ch.9]. Standard loss functions, such as squared error or Huber's loss satisfy these conditions, hence the corresponding estimators are polynomial-time.

Unfortunately, loss minimization with a (non-constant) convex loss yields unbounded response to even a single outlier [3, Ch.5]. We extend this result to also account for regularization and RKHSs.

**Theorem 1.** *Empirical risk minimization based on a (non-constant) convex loss cannot have bounded response if the domain (or kernel) is unbounded, even under Tikhonov regularization.* (Proof given in Appendix B of the supplement.)

By contrast, consider the case of a (non-constant) *bounded* loss function.[6] Bounded loss functions are a common choice in robust regression because they not only ensure bounded response, trivially, they can also ensure a high breakdown point of $(n-p)/(2n)$ [3, Ch.5]. Unfortunately, estimators based on bounded losses are inherently intractable.

**Theorem 2.** *Bounded (non-constant) loss minimization is NP-hard.* (Proof given in Appendix E.)

These difficulties with empirical risk minimization have led the field of robust statistics to develop a variety of alternative estimators [4, Ch.7]. For example, [7] recommends subset-selection based regression estimators, such as Least Trimmed Loss:

$$\hat{\boldsymbol{\theta}}_{LTL} \quad \in \quad \arg\min_{\boldsymbol{\theta}\in\Theta} \sum_{i=1}^{n'} \rho(r_{[i]}). \tag{8}$$

Here $r_{[i]}$ denotes sorted residuals $r_{[1]} \leq \cdots \leq r_{[n]}$ and $n' < n$ is the number of terms to consider. Traditionally $\rho(r) = r^2$ is used. These estimators are known to have high breakdown [7],[7] and obviously demonstrate bounded response to single outliers. Unfortunately, (8) is NP-hard [1].

## 3 Variational M-estimation

To address the dilemma, we first adopt a general form of adaptive M-estimator that allows flexibility while allowing a general approximation strategy. The key construction is a variational representation of M-estimation that can express a number of standard robust (and non-robust) methods in a common framework. In particular, consider the following adaptive form of loss function

$$\rho(r) \quad = \quad \min_{0\leq\eta\leq1} \eta\ell(r) + \psi(\eta). \tag{9}$$

where $r$ is a residual value, $\ell$ is a closed *convex* base loss, $\eta$ is an adaptive weight on the base loss, and $\psi$ is a convex auxiliary function. The weight can choose to ignore the base loss if $\ell(r)$ is large, but this is balanced against a prior penalty $\psi(\eta)$. Different choices of base loss and auxiliary function will yield different results, and one can represent a wide variety of loss functions $\rho$ in this way [8]. For example, any convex loss $\rho$ can be trivially represented in the form (9) by setting $\ell = \rho$, and $\psi(\eta) = \delta_{\{1\}}(\eta)$.[8] *Bounded* loss functions can also be represented in this way, for example

| | | | |
|---|---|---|---|
| (Geman-McClure) [8] | $\rho(r) = \frac{r^2}{1+r^2}$ | $\ell(r) = r^2$ $\quad$ $\psi(\eta) = (\sqrt{\eta} - 1)^2$ | (10) |
| (Geman-Reynolds) [8] | $\rho(r) = \frac{|r|}{1+|r|}$ | $\ell(r) = |r|$ $\quad$ $\psi(\eta) = (\sqrt{\eta} - 1)^2$ | (11) |
| (LeClerc) [8] | $\rho(r) = 1 - \exp(-\ell(r))$ | $\ell(\cdot)$ convex $\quad$ $\psi(\eta) = \eta\log\eta - \eta + 1$ | (12) |
| (Clipped-loss) [9] | $\rho(r) = \max(1, \ell(r))$ | $\ell(\cdot)$ convex $\quad$ $\psi(\eta) = 1 - \eta$. | (13) |

Appendix D in the supplement demonstrates how one can represent general functions $\rho$ in the form (9), not just specific examples, significantly extending [8] with a general characterization.

Therefore, all of the previous forms of regularized empirical risk minimization, whether with a convex or bounded loss $\rho$, can be easily expressed using only convex base losses $\ell$ and convex auxiliary functions $\psi$, as follows

$$\hat{\boldsymbol{\theta}}_{VM} \quad \in \quad \arg\min_{\boldsymbol{\theta}\in\Theta} \min_{0\le\boldsymbol{\eta}\le 1} \boldsymbol{\eta}^T\boldsymbol{\ell}(\mathbf{y}-X\boldsymbol{\theta}) + \mathbf{1}^T\boldsymbol{\psi}(\boldsymbol{\eta}) + \tfrac{\lambda}{2}\|\boldsymbol{\eta}\|_1\|\boldsymbol{\theta}\|_2^2 \qquad (14)$$

$$\hat{f}_{VM} \quad \in \quad \arg\min_{f\in\mathcal{H}} \min_{0\le\boldsymbol{\eta}\le 1} \sum_{i=1}^n \{\eta_i\ell(y_i-f(x_i)) + \psi(\eta_i)\} + \tfrac{\lambda}{2}\|\boldsymbol{\eta}\|_1\|f\|_{\mathcal{H}}^2 \qquad (15)$$

$$\hat{\boldsymbol{\alpha}}_{VM} \quad \in \quad \arg\min_{\boldsymbol{\alpha}} \min_{0\le\boldsymbol{\eta}\le 1} \boldsymbol{\eta}^T\boldsymbol{\ell}(\mathbf{y}-K\boldsymbol{\alpha}) + \mathbf{1}^T\boldsymbol{\psi}(\boldsymbol{\eta}) + \tfrac{\lambda}{2}\|\boldsymbol{\eta}\|_1\boldsymbol{\alpha}^T K\boldsymbol{\alpha}. \qquad (16)$$

Note that we have added a regularizer $\|\boldsymbol{\eta}\|_1/n$, which increases robustness by encouraging $\eta$ weights to prefer small values (but adaptively increase on indices with small loss). This particular form of regularization has two advantages: (i) it is a smooth function of $\boldsymbol{\eta}$ on $0 \le \boldsymbol{\eta} \le 1$ (since $\|\boldsymbol{\eta}\|_1 = \mathbf{1}^T\boldsymbol{\eta}$ in this case), and (ii) it enables a tight convex approximation strategy, as we will see below.

Note that other forms of robust regression can be expressed in a similar framework. For example, generalized M-estimation (GM-estimation) can be formulated simply by forcing each $\eta_i$ to take on a specific value determined by $\|x_i\|$ or $r_i$ [7], ignoring the auxilary function $\psi$. Least Trimmed Loss (8) can be expressed in the form (9) provided only that we add a shared constraint over $\boldsymbol{\eta}$:

$$\hat{\boldsymbol{\theta}}_{LTL} \quad \in \quad \arg\min_{\boldsymbol{\theta}\in\Theta} \min_{0\le\boldsymbol{\eta}\le 1:\mathbf{1}^T\boldsymbol{\eta}=n'} \boldsymbol{\eta}^T\boldsymbol{\ell}(\mathbf{r}) + \boldsymbol{\psi}(\boldsymbol{\eta}) \qquad (17)$$

where $\psi(\eta_i) = 1 - \eta_i$ and $n' < n$ specifies the number of terms to consider in the sum of losses. Since $\boldsymbol{\eta} \in \{0,1\}^n$ at a solution (see e.g. [9]), (17) is equivalent to (8) if $\psi$ is the clipped loss (13).

These formulations are all convex in the parameters given the auxiliary weights, and vice versa. However, they are not jointly convex in the optimization variables (i.e. in $\boldsymbol{\theta}$ and $\boldsymbol{\eta}$, or in $\boldsymbol{\alpha}$ and $\boldsymbol{\eta}$). Therefore, one is not assured that the problems (14)–(16) have only global minima; in fact local minima exist and global minima cannot be easily found (or even verified).

## 4 Computationally Efficient Approximation

We present a general approximation strategy for the variational regression estimators above that can guarantee polynomial run-time while ensuring certain robustness and consistency properties. The approximation is significantly tighter than the existing work [9], which allows us to achieve stronger guarantees while providing better empirical performance. In developing our estimator we follow standard methodology from combinatorial optimization: given an intractable optimization problem, first formulate a (hopefully tight) convex relaxation that provides a lower bound on the objective, then round the relaxed minimizer back to the feasible space, hopefully verifying that the rounded solution preserves desirable properties, and finally re-optimize the rounded solution to refine the result; see e.g. [23].

To maintain generality, we formulate the approximate estimator in the RKHS setting. Consider (16). Although the problem is obviously convex in $\boldsymbol{\alpha}$ given $\boldsymbol{\eta}$, and vice versa, it is not jointly convex (recall the assumption that $\ell$ and $\psi$ are both convex functions). This suggests that an obvious computational strategy for computing the estimator (16) is to alternate between $\boldsymbol{\alpha}$ and $\boldsymbol{\eta}$ optimizations (or use heuristic methods [2]), but this cannot guarantee anything other than local solutions (and thus may not even achieve any of the desired theoretical properties associated with the estimator).

***Reformulation:*** We first need to reformulate the problem to allow a tight relaxation. Let $\Delta(\boldsymbol{\eta})$ denote putting a vector $\boldsymbol{\eta}$ on the main diagonal of a square matrix, and let $\circ$ denote componentwise multiplication. Since $\ell$ is closed and convex by assumption, we know that $\ell(r) = \sup_\nu \nu r - \nu\ell^*(\nu)$, where $\ell^*$ is the Fenchel conjugate of $\ell$ [22]. This allows (16) to be reformulated as follows.

**Lemma 1.** $\min_{0\le\boldsymbol{\eta}\le 1} \min_{\boldsymbol{\alpha}} \boldsymbol{\eta}^T\boldsymbol{\ell}(\mathbf{y}-K\boldsymbol{\alpha}) + \mathbf{1}^T\boldsymbol{\psi}(\boldsymbol{\eta}) + \tfrac{\lambda}{2}\|\boldsymbol{\eta}\|_1\boldsymbol{\alpha}^T K\boldsymbol{\alpha}$     (18)

$$= \min_{0\le\boldsymbol{\eta}\le 1} \sup_{\boldsymbol{\nu}} \mathbf{1}^T\boldsymbol{\psi}(\boldsymbol{\eta}) - \boldsymbol{\eta}^T(\boldsymbol{\ell}^*(\boldsymbol{\nu}) - \Delta(\mathbf{y})\boldsymbol{\nu}) - \tfrac{1}{2\lambda}\boldsymbol{\nu}^T\left(K\circ(\boldsymbol{\eta}\|\boldsymbol{\eta}\|_1^{-1}\boldsymbol{\eta}^T)\right)\boldsymbol{\nu}, \quad (19)$$

*where the function evaluations are componentwise.* (Proof given in Appendix A of the supplement.)

Although no relaxation has been introduced, the new form (25) has a more convenient structure.

***Relaxation:*** Let $N = \boldsymbol{\eta}\|\boldsymbol{\eta}\|_1^{-1}\boldsymbol{\eta}^T$ and note that, since $0 \leq \boldsymbol{\eta} \leq 1$, $N$ must satisfy a number of useful properties. We can summarize these by formulating a constraint set $N \in \mathcal{N}_{\boldsymbol{\eta}}$ given by:

$$\mathcal{N}_{\boldsymbol{\eta}} = \{N : N \succcurlyeq 0, N\mathbf{1} = \boldsymbol{\eta}, \mathrm{rank}(N) = 1\} \tag{20}$$

$$\mathcal{M}_{\boldsymbol{\eta}} = \{M : M \succcurlyeq 0, M\mathbf{1} = \boldsymbol{\eta}, \mathrm{tr}(M) \leq 1\}. \tag{21}$$

Unfortunately, the set $\mathcal{N}_{\boldsymbol{\eta}}$ is not convex because of the rank constraint. However, relaxing this constraint leads to a set $\mathcal{M}_{\boldsymbol{\eta}} \supseteq \mathcal{N}_{\boldsymbol{\eta}}$ which preserves much of the key structure, as we verify below.

**Lemma 2.** $(25) = \min_{0 \leq \boldsymbol{\eta} \leq 1} \min_{N \in \mathcal{N}_{\boldsymbol{\eta}}} \sup_{\boldsymbol{\nu}} \mathbf{1}^T \boldsymbol{\psi}(\boldsymbol{\eta}) - \boldsymbol{\eta}^T(\boldsymbol{\ell}^*(\boldsymbol{\nu}) - \Delta(\mathbf{y})\boldsymbol{\nu}) - \frac{1}{2\lambda}\boldsymbol{\nu}^T(K \circ N)\boldsymbol{\nu} \tag{22}$

$$\geq \min_{0 \leq \boldsymbol{\eta} \leq 1} \min_{M \in \mathcal{M}_{\boldsymbol{\eta}}} \sup_{\boldsymbol{\nu}} \mathbf{1}^T \boldsymbol{\psi}(\boldsymbol{\eta}) - \boldsymbol{\eta}^T(\boldsymbol{\ell}^*(\boldsymbol{\nu}) - \Delta(\mathbf{y})\boldsymbol{\nu}) - \frac{1}{2\lambda}\boldsymbol{\nu}^T(K \circ M)\boldsymbol{\nu}. \tag{23}$$

*using the fact that* $\mathcal{N}_{\boldsymbol{\eta}} \subseteq \mathcal{M}_{\boldsymbol{\eta}}$. *(Proof given in Appendix A of the supplement.)*

Crucially, the constraint set $\{(\boldsymbol{\eta}, M) : 0 \leq \boldsymbol{\eta} \leq 1, M \in \mathcal{M}_{\boldsymbol{\eta}}\}$ is jointly convex in $\boldsymbol{\eta}$ and $M$, thus (35) is a convex-concave min-max problem. To see why, note that the inner objective function is jointly convex in $\boldsymbol{\eta}$ and $M$, and concave in $\boldsymbol{\nu}$. Since a pointwise maximum of convex functions is convex, the problem is convex in $(\boldsymbol{\eta}, M)$ [22, Ch.3]. We conclude that all local minima in $(\boldsymbol{\eta}, M)$ are global. Therefore, (35) provides the foundation for an efficiently solvable relaxation.

***Rounding:*** Unfortunately the solution to $M$ in (35) does not allow direct recovery of an estimator $\boldsymbol{\alpha}$ achieving the same objective value in (24), unless $M$ satisfies $\mathrm{rank}(M) = 1$. In general we first need to round $M$ to a rank 1 solution. Fortunately, a trivial rounding procedure is available: we simply use $\boldsymbol{\eta}$ (ignoring $M$) and re-solve for $\boldsymbol{\alpha}$ in (24). This is equivalent to replacing $M$ with the rank 1 matrix $\tilde{N} = \boldsymbol{\eta}\|\boldsymbol{\eta}\|_1^{-1}\boldsymbol{\eta}^T \in \mathcal{N}_{\boldsymbol{\eta}}$, which restores feasibility in the original problem. Of course, such a rounding step will generally increase the objective value.

***Reoptimization:*** Finally, the rounded solution can be locally improved by alternating between $\boldsymbol{\eta}$ and $\boldsymbol{\alpha}$ updates in (24) (or using any other local optimization method), yielding the final estimate $\tilde{\boldsymbol{\alpha}}$.

# 5 Properties

Although a tight *a priori* bound on the size of the optimality gap is difficult to achieve, a rigorous bound on the optimality gap can be recovered *post hoc* once the re-optimized estimator is computed. Let $R_0$ denote the minimum value of (24) (not efficiently computable); let $R_1$ denote the minimum value of (35) (the relaxed solution); let $R_2$ denote the value of (24) achieved by freezing $\boldsymbol{\eta}$ from the relaxed solution but re-optimizing $\boldsymbol{\alpha}$ (the rounded solution); and finally let $R_3$ denote the value of (24) achieved by re-optimizing $\boldsymbol{\eta}$ and $\boldsymbol{\alpha}$ from the rounded solution (the re-optimized solution). Clearly we have the relationships $R_1 \leq R_0 \leq R_3 \leq R_2$. An upper bound on the relative optimality gap of the final solution ($R_3$) can be determined by $(R_3 - R_0)/R_3 \leq (R_3 - R_1)/R_3$, since $R_1$ and $R_3$ are both known quantities.

***Tractability:*** Under mild assumptions on $\ell$ and $\psi$, computation of the approximate estimator (solving the relaxed problem, rounding, then re-optimizing) admits a polynomial-time solution; see Appendix E in the supplement. (Appendix E also provides details for an efficient implementation for solving (35).) Once $\boldsymbol{\eta}$ is recovered from the relaxed solution, the subsequent optimizations of (24) can be solved efficiently under weak assumptions about $\ell$ and $\psi$; namely that they both satisfy the self-concordance and polynomial-time computation properties discussed in Section 2.

***Robustness:*** Despite the approximation, the relaxation remains sufficiently tight to preserve some of the robustness properties of bounded loss minimization. To establish the robustness (and consistency) properties, we will need to make use of a specific technical definition of *outliers* and *inliers*.

**Definition 2** (Outliers and Inliers). *For an $L$-Lipschitz loss $\ell$, an outlier is a point $(x_i, y_i)$ that satisfies $\ell(y_i) > L^2 K_{ii}/(2\lambda) - \psi'(0)$, while an inlier satisfies $\ell(y_i) + L^2 K_{ii}/(2\lambda) < -\psi'(1)$.*

**Theorem 3.** *Assume the loss $\rho$ is bounded and has a variational representation (9) such that $\ell$ is Lipschitz-continuous and $\psi'$ is bounded. Also assume there is at least one (unperturbed) inlier, and consider the perturbation of a single data point $(y_1, x_1)$. Under the following conditions, the rounded (re-optimized) estimator maintains bounded response:*
*(i) If either $y_1$ remains bounded, or $\kappa(x_1, x_1)$ remains bounded.*
*(ii) If $|y_1| \to \infty$, $\kappa(x_1, x_1) \to \infty$ and $\ell(y_1)/\kappa(x_1, x_1) \to \infty$.*
*(Proof given in Appendix B of the supplement.)*

| Methods | Outlier Probability | | | | | |
|---|---|---|---|---|---|---|
| | $p = 0.4$ | | $p = 0.2$ | | $p = 0.0$ | |
| L2 | 43.5 | (13) | 57.6 | (21.21) | 0.52 | (0.01) |
| L1 | 4.89 | (2.81) | 3.6 | (2.04) | 0.52 | (0.01) |
| Huber | 4.89 | (2.81) | 3.62 | (2.02) | 0.52 | (0.01) |
| LTS | 6.72 | (7.37) | 8.65 | (14.11) | 0.52 | (0.01) |
| GemMc | 0.53 | (0.03) | 0.52 | (0.02) | 0.52 | (0.01) |
| [9] | 0.52 | (0.01) | 0.52 | (0.01) | 0.52 | (0.01) |
| AltBndL2 | 0.52 | (0.01) | 0.52 | (0.01) | 0.52 | (0.02) |
| AltBndL1 | 0.73 | (0.12) | 0.74 | (0.16) | 0.52 | (0.01) |
| CvxBndL2 | 0.52 | (0.01) | 0.52 | (0.01) | 0.52 | (0.01) |
| CvxBndL1 | 0.53 | (0.02) | 0.55 | (0.05) | 0.52 | (0.01) |

Table 1: RMSE on clean test data for an artificial data set with 5 features and 100 training points, with outlier probability $p$, and 10000 test data points. Results are averaged over 10 repetitions. Standard deviations are given in parentheses.

Note that the latter condition causes *any* convex loss $\ell$ to demonstrate unbounded response (see proof of Theorem 5 in Appendix B). Therefore, the approximate estimator is strictly more robust (in terms of bounded response) than regularized empirical risk minimization with a convex loss $\ell$.

***Consistency:*** Finally, we can establish consistency of the approximate estimator in a limited albeit non-trivial setting, although we have yet to establish it generally.

**Theorem 4.** *Assume $\ell$ is Lipschitz-continuous and $\psi(\eta) = 1 - \eta$. Assume that the data is generated from a mixture of* inliers *and* outliers*, where $P(inlier) > P(outlier)$. Then the estimate $\hat{\boldsymbol{\theta}}$ produced by the rounded (re-optimized) method is loss consistent.*(Proof given in Appendix C.2.)

## 6 Experimental Evaluation

We conducted a set of experiments to evaluate the effectiveness of the proposed method compared to standard methods from the literature. Our experimental evaluation was conducted in two parts: first a synthetic experiment where we could control data generation, then an experiment on real data.

The first synthetic experiment was conducted as follows. A target weight vector $\boldsymbol{\theta}$ was drawn from $N(\mathbf{0}, I)$, with $X_{i:}$ sampled uniformly from $[0, 1]^m$, $m = 5$, and outputs $y_i$ computed as $y_i = X_{i:}\boldsymbol{\theta} + \epsilon_i$, $\epsilon_i \sim N(0, \frac{1}{2})$. We then seeded the data set with outliers by randomly re-sampling each $y_i$ and $X_{i:}$ from $N(0, 10^8)$ and $N(0, 10^4)$ respectively, governed by an outlier probability $p$. Then we randomly sampled 100 points as the training set and another 10000 samples are used for testing.

We implemented the proposed method with two different base losses, $L_2$ and $L_1$, respectively; referring to these as CvxBndL2 and CvxBndL1. We compared to standard $L_2$ and $L_1$ loss minimization, as well as minimizing the Huber minimax loss (Huber) [4]. We also considered standard methods from the robust statistics literature, including the least trimmed square method (LTS) [7, 24], and bounded loss minimization based on the Geman-McClure loss (GemMc) [8]. Finally we also compared to the alternating minimization strategies outlined at the end of Section 3 (AltBndL2 and AltBndL1 for $L_2$ and $L_1$ losses respectively), and implemented the strategy described in [9]. We added the Tikhonov regularization to each method and the regularization parameter $\lambda$ was selected (optimally for each method) on a separate validation set. Note that LTS has an extra parameter $n'$, which is the number of inliers. The ideal setting $n' = (1 - p)n$ was granted to LTS. We also tried 30 random restarts for LTS and picked the best result.

All experiments are repeated 10 times and the average root mean square errors (RMSE) (with standard deviations) on the clean test data are reported in Table 1. For $p = 0$ (*i.e.* no outliers), all methods perform well; their RMSEs are close to optimal (1/2, the standard deviation of $\epsilon_i$). However, when outliers start to appear, the result of least squares is significantly skewed, while the results of classic robust statistics methods, Huber, L1 and LTS, indeed turn out to be more robust than the least squares, but nevertheless are still affected significantly. Both implementations of the new method performs comparably to the the non-convex Geman-McClure loss while substantially improving the alternating strategy under the L1 loss. Note that the latter improvement clearly demonstrates that

| Methods | Datasets | | | | | | | |
|---|---|---|---|---|---|---|---|---|
| | cal-housing | | abalone | | pumadyn | | bank-8fh | |
| L2 | 1185 | (124.59) | 7.93 | (0.67) | 1.24 | (0.42) | 18.21 | (6.57) |
| L1 | 1303 | (244.85) | 7.30 | (0.40) | 1.29 | (0.42) | 6.54 | (3.09) |
| Huber | 1221 | (119.18) | 7.73 | (0.49) | 1.24 | (0.42) | 7.37 | (3.18) |
| LTS | 533 | (398.92) | 755.1 | (126) | 0.32 | (0.41) | 10.96 | (6.67) |
| GemMc | 28 | (88.45) | 2.30 | (0.01) | 0.12 | (0.12) | 0.93 | (0.80) |
| [9] | 967 | (522.40) | 8.39 | (0.54) | 0.81 | (0.77) | 3.91 | (6.18) |
| AltBndL2 | 967 | (522.40) | 8.39 | (0.54) | 0.81 | (0.77) | 7.74 | (9.40) |
| AltBndL1 | 1005 | (603.00) | 7.30 | (0.40) | 1.29 | (0.42) | 1.61 | (2.51) |
| CvxBndL2 | 9 | (0.64) | 7.60 | (0.86) | 0.07 | (0.07) | 0.20 | (0.05) |
| CvxBndL1 | 8 | (0.28) | 2.98 | (0.08) | 0.08 | (0.07) | 0.10 | (0.07) |
| Gap(Cvx2) | 2e-12 | (3e-12) | 3e-9 | (4e-9) | 0.025 | (0.052) | 0.001 | (0.003) |
| Gap(Cvx1) | 0.005 | (0.01) | 0.001 | (0.001) | 0.267 | (0.269) | 0.011 | (0.028) |

Table 2: RMSE on clean test data for 108 training data points and 1000 test data points, with 10 repeats. Standard deviations shown parentheses. The mean gap values of CvxBndL2 and CvxBndL1, Gap(Cvx2) and Gap(Cvx1) respectively, are given in the last two rows.

alternating can be trapped in poor local minima. The proposal from [9] was not effective in this setting (which differed from the one investigated there).

Next, we conducted an experiment on four real datasets taken from the StatLib repository[9] and DELVE.[10] For each data set, we randomly selected 108 points as the training set, and another random 1000 points as the test set. Here the regularization constant is tuned by 10-fold cross validation. To seed outliers, 5% of the training set are randomly chosen and their $X$ and $y$ values are multiplied by 100 and 10000, respectively. All of these data sets have 8 features, except pumadyn which has 32 features. We also estimated the scale factor on the training set by the mean absolute deviation method, a common method in robust statistics [3]. Again, the ideal parameter $n' = (1 - 5\%)n$ is granted to LTS and 30 random restarts are performed.

The RMSE on test set for all methods are reported in Table 2. It is clear that all methods based on convex losses (L2, L1, Huber) suffer significantly from the added outliers. The method proposed in this paper consistently outperform all other methods with a noticeable margin, except on the abalone data set where GemMc performs slightly better.[11] Again, we observe evidence that the alternating strategy can be trapped in poor local minima, while the method from [9] was less effective. We also measured the relative optimality gaps for the approximate CvxBnd procedures. The gaps were quite small in most cases (the gaps were very close to zero in the synthetic case, and so are not shown), demonstrating the tightness of the proposed approximation scheme.

## 7 Conclusion

We have developed a new robust regression method that can guarantee a form of robustness (bounded response) while ensuring tractability (polynomial run-time). The estimator has been proved consistent under some restrictive but non-trivial conditions, although we have not established general consistency. Nevertheless, an empirical evaluation reveals that the method meets or surpasses the generalization ability of state-of-the-art robust regression methods in experimental studies. Although the method is more computationally involved than standard approaches, it achieves reasonable scalability in real problems. We are investigating whether the proposed estimator achieves stronger robustness properties, such as high breakdown or bounded influence. It would be interesting to extend the approach to also estimate scale in a robust and tractable manner. Finally, we continue to investigate whether other techniques from the robust statistics and machine learning literatures can be incorporated in the general framework while preserving desired properties.

**Acknowledgements**

Research supported by AICML and NSERC.

## Footnotes

[1] Generally one has to introduce an additional scale parameter $\sigma$ and allow rescaling of the residuals via $r_i/\sigma$, to preserve parameter equivariance [3, 4]. However, we will initially assume a known scale.

[2] We are obviously assuming $\mathcal{X}$ is equipped with an appropriate $\sigma$-algebra, and $\mathbb{R}$ with the standard Borel $\sigma$-algebra, such that the joint distribution $P$ over $\mathcal{X} \times \mathbb{R}$ is well defined and $\kappa(\cdot, \cdot)$ is measurable.

[3] In particular, let $M_n(\boldsymbol{\theta}) = \frac{1}{n} \sum_{i=1}^{n} \rho(y_i - \mathbf{x}_i^T \boldsymbol{\theta})$, let $M(\boldsymbol{\theta}) = \mathbb{E}(\rho(y_1 - \mathbf{x}_1^T \boldsymbol{\theta}))$, and equip the parameter space $\Theta$ with the uniform metric $\| \cdot \|_{\Theta}$. Then $\hat{\boldsymbol{\theta}}_M^{(n)} \to \boldsymbol{\theta}^*$, provided $\|M_n - M\|_{\Theta} \to 0$ in outer probability (adopted to avoid measurability issues) and $M(\boldsymbol{\theta}^*) > \sup_{\boldsymbol{\theta} \in G} M(\boldsymbol{\theta})$ for every open set $G$ that contains $\boldsymbol{\theta}^*$. The latter assumption is satisfied in particular when $M : \Theta \mapsto \mathbb{R}$ is upper semicontinuous with a unique maximum at $\boldsymbol{\theta}^*$. It is also possible to derive asymptotic convergence rates for general $M$-estimators [16].

[4] Specifically, let $\rho^* = \inf_{f \in \mathcal{H}} E[\rho(y_1 - f(x_1))]$. Then [6] showed that $\frac{1}{n} \sum_{i=1}^{n} \rho(y_i - \hat{f}_{MR}(x_i)) \to \rho^*$ provided the regularization constant $\lambda_n \to 0$ and $\lambda_n^2 n \to \infty$, the loss $\rho$ is convex and Lipschitz-continuous, and the RKHS $\mathcal{H}$ (induced by some bounded measurable kernel $\kappa$) is separable and dense in $L_1(\mathbb{P})$ (the space of $\mathbb{P}$-integrable functions) for all distributions $\mathbb{P}$ on $\mathcal{X}$. Also, $\mathcal{Y} \subset \mathbb{R}$ is required to be *closed* where $y \in \mathcal{Y}$.

[5] A function $\rho$ is *self-concordant* if $|\rho'''(r)| \leq 2\rho''(r)^{3/2}$; see e.g. [22, Ch.9].

[6] A bounded function obviously cannot be convex over an unbounded domain unless it is constant.

[7] When $n'$ approaches $n/2$ the breakdown of (8) approaches $1/2$ [7].

[8] We use $\delta_C(\eta)$ to denote the indicator for the point set $C$; i.e., $\delta_C(\eta) = 0$ if $\eta \in C$, otherwise $\delta_C(\eta) = \infty$.

[9] http://lib.stat.cmu.edu/datasets/

[10] http://www.cs.utoronto.ca/ delve/data/summaryTable.html

[11] Note that we obtain different results than [9] arising from a very different outlier process.

# References

[1] T. Bernholt. Robust estimators are hard to compute. Technical Report 52/2005, SFB475, U. Dortmund, 2005.

[2] R. Nunkesser and O. Morell. An evolutionary algorithm for robust regression. *Computational Statistics and Data Analysis*, 54:3242–3248, 2010.

[3] R. Maronna, R. Martin, and V. Yohai. *Robust Statistics: Theory and Methods*. Wiley, 2006.

[4] P. Huber and E. Ronchetti. *Robust Statistics*. Wiley, 2nd edition, 2009.

[5] A. Christmann and I. Steinwart. Consistency and robustness of kernel-based regression in convex risk minimization. *Bernoulli*, 13(3):799–819, 2007.

[6] A. Christmann, A. Van Messem, and I. Steinwart. On consistency and robustness properties of support vector machines for heavy-tailed distributions. *Statistics and Its Interface*, 2:311–327, 2009.

[7] P. Rousseeuw and A. Leroy. *Robust Regression and Outlier Detection*. Wiley, 1987.

[8] M. Black and A. Rangarajan. On the unification of line processes, outlier rejection, and robust statistics with applications in early vision. *International Journal of Computer Vision*, 19(1): 57–91, 1996.

[9] Y. Yu, M. Yang, L. Xu, M. White, and D. Schuurmans. Relaxed clipping: A global training method for robust regression and classification. In *Advances in Neural Information Processings Systems (NIPS)*, 2010.

[10] A. Bental, L. El Ghaoui, and A. Nemirovski. *Robust Optimization*. Princeton Series in Applied Mathematics. Princeton University Press, October 2009.

[11] H. Xu, C. Caramanis, and S. Mannor. Robust regression and Lasso. In *Advances in Neural Information Processing Systems (NIPS)*, volume 21, pages 1801–1808, 2008.

[12] H. Xu, C. Caramanis, and S. Mannor. Robustness and regularization of support vector machines. *Journal of Machine Learning Research*, 10:1485–1510, 2009.

[13] S. Mukherjee, P. Niyogi, T. Poggio, and R. Rifkin. Learning theory: Stability is sufficient for generalization and necessary and sufficient for consistency of empirical risk minimization. *Advances in Computational Mathematics*, 25(1-3):161–193, 2006.

[14] G. Kimeldorf and G. Wahba. A correspondence between Bayesian estimation on stochastic processes and smoothing by splines. *Annals of Mathematical Statistics*, 41(2):495–502, 1970.

[15] I. Steinwart and A. Christmann. *Support Vector Machines*. Springer, 2008.

[16] Aad W. van der Vaart and Jon A. Wellner. *Weak Convergence and Empirical Processes*. Springer, 1996.

[17] F. Hampel, E. Ronchetti, P. Rousseeuw, and W. Stahel. *Robust Statistics: The Approach Based on Influence Functions*. Wiley, 1986.

[18] D. Donoho and P. Huber. The notion of breakdown point. In *A Festschrift for Erich L. Lehmann*, pages 157–184. Wadsworth, 1983.

[19] P. Davies and U. Gather. The breakdown point—examples and counterexamples. *REVSTAT Statistical Journal*, 5(1):1–17, 2007.

[20] Y. Nesterov and A. Nemiroviskii. *Interior-point Polynomial Methods in Convex Programming*. SIAM, 1994.

[21] Y. Nesterov. *Introductory Lectures on Convex Optimization: A Basic Course*. Kluwer, 2003.

[22] S. Boyd and L. Vandenberghe. *Convex Optimization*. Cambridge U. Press, 2004.

[23] J. Peng and Y. Wei. Approximating k-means-type clustering via semidefinite programming. *SIAM Journal on Optimization*, 18(1):186–205, 2007.

[24] P. Rousseeuw and K. Van Driessen. Computing LTS regression for large data sets. *Data Mining and Knowledge Discovery*, 12(1):29–45, 2006.

[25] R. Horn and C. Johnson. *Matrix Analysis*. Cambridge, 1985.

